# 3D Object Recognition Using Unsupervised Feature Extraction

**Nathan Intrator**
Center for Neural Science,
Brown University
Providence, RI 02912, USA

**Josh I. Gold**
Center for Neural Science,
Brown University
Providence, RI 02912, USA

**Heinrich H. Bülthoff**
Dept. of Cognitive Science,
Brown University,
and Center for
Biological Information Processing,
MIT, Cambridge, MA 02139 USA

**Shimon Edelman**
Dept. of Applied Mathematics
and Computer Science,
Weizmann Institute of Science,
Rehovot 76100, Israel

## Abstract

Intrator (1990) proposed a feature extraction method that is related to recent statistical theory (Huber, 1985; Friedman, 1987), and is based on a biologically motivated model of neuronal plasticity (Bienenstock et al., 1982). This method has been recently applied to feature extraction in the context of recognizing 3D objects from single 2D views (Intrator and Gold, 1991). Here we describe experiments designed to analyze the nature of the extracted features, and their relevance to the theory and psychophysics of object recognition.

## 1  Introduction

Results of recent computational studies of visual recognition (e.g., Poggio and Edelman, 1990) indicate that the problem of recognition of 3D objects can be effectively reformulated in terms of standard pattern classification theory. According to this approach, an object is represented by a few of its 2D views, encoded as clusters in multidimentional space. Recognition of a novel view is then carried out by interpolating among the stored views in the representation space. A major characteristic of the view interpolation scheme is its sensitivity to viewpoint: the farther the novel view is from the stored views, the lower the expected recognition rate.

This characteristic performance in the recognition of novel views of synthetic 3D stimuli was indeed found in human subjects by Bülthoff and Edelman (1991), who also replicated it in simulated psychophysical experiments that involved a computer implementation of the view interpolation model. Because of the high dimensionality of the raster images seen by the human subjects, it was impossible to use them directly for classification in the simulated experiments. Consequently, the simulations were simplified, in that the views presented to the model were encoded as lists of vertex locations of the objects (which resembled 3D wire frames).

This simplification amounts to what is referred to in the psychology of recognition as the feature extraction step (LaBerge, 1976). The discussion of the issue of features of recognition in recent psychological literature is relatively scarce, probably because of the abandonment of invariant feature theories (which postulate that objects are represented by clusters of points in multidimensional feature spaces (Duda and Hart, 1973)) in favor of structural models (see review in (Edelman, 1991)). Although some attempts have been made to generate and verify specific psychophysical predictions based on the feature space approach (see especially (Shepard, 1987)), current feature-based theories of perception seem to be more readily applicable to lower-level visual tasks than to the problem of object recognition.

In the present work, our aim was to explore a computationally tractable model of feature extraction conceived as dimensionality reduction, and to test its psychophysical validity. This work was guided by previous successful applications in pattern recognition of dimensionality reduction by a network model implementing Exploratory Projection Pursuit (Intrator, 1990; Intrator and Gold, 1991). We were also motivated by results of recent psychophysical experiments (Edelman and Bülthoff, 1990; Edelman et al., 1991) that found improvement in subjects' performance with increasing stimulus familiarity. These results are compatible with a feature-based recognition model which extracts problem-specific features in addition to universal ones. Specifically, the subjects' ability to discern key elements of the solution appears to increase as the problem becomes more familiar. This finding suggests that some of the features used by the visual system are based on the task-specific data, and therefore raises the question of how can such features be extracted. It was our conjecture that features found by the EPP model would turn out to be similar to the task-specific features in human vision.

## 1.1   Unsupervised Feature Extraction - The BCM Model

The feature extraction method briefly described below emphasizes dimensionality reduction, while seeking features of a set of objects that would best distinguish among the members of the set. This method does not rely on a general pre-defined set of features. This is not to imply, however, that the features are useful only in recognition of the original set of images from which they were extracted. In fact, the potential importance of these features is related to their invariance properties, or their ability to generalize. Invariance properties of features extracted by this method have been demonstrated previously in speech recognition (Intrator and Tajchman,

1991; Intrator, 1992).

From a mathematical viewpoint, extracting features from gray level images is related to dimensionality reduction in a high dimensional vector space, in which an $n \times k$ pixel image is considered to be a vector of length $n \times k$. The dimensionality reduction is achieved by replacing each image (or its high dimensional equivalent vector) by a low dimensional vector in which each element represents a projection of the image onto a vector of synaptic weights (constructed by a BCM neuron).

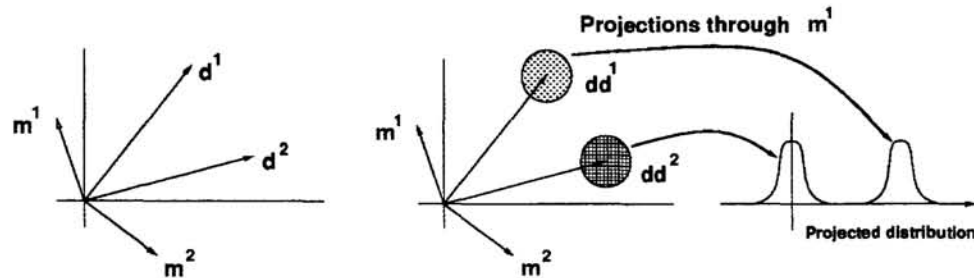

Figure 1: The stable solutions for a two dimensional two input problem are $m_1$ and $m_2$ (left) and similarly with a two-cluster data (right).

The feature extraction method we used (Intrator and Cooper, 1991) seeks multi-modality in the projected distribution of these high dimensional vectors. A simple example is illustrated in Figure 1. For a two-input problem in two dimensions, the stable solutions (projection directions) are $m_1$ and $m_2$, each has the property of being orthogonal to one of the inputs. In a higher dimensional space, for $n$ linearly independent inputs, a stable solution is one that it is orthogonal to all but one of the inputs. In case of noisy but clustered inputs, a stable solution will be orthogonal to all but one of the cluster centers. As is seen in Figure 1 (right), this leads to a bimodal, or, in general, multi-modal, projected distribution. Further details are given in (Intrator and Cooper, 1991). In the present study, the features extracted by the above approach were used for classification as described in (Intrator and Gold, 1991; Intrator, 1992).

## 1.2    Experimental paradigm

We have studied the features extracted by the BCM model by replicating the experiments of Bülthoff and Edelman (1991), designed to test generalization from familiar to novel views of 3D objects. As in the psychophysical experiments, images of novel wire-like computer-generated objects (Bülthoff and Edelman, 1991; Edelman and Bülthoff, 1990) were used as stimuli. These objects proved to be easily manipulated, and yet complex enough to yield interesting results. Using wires also simplified the problem for the feature extractor, as they provided little or no occlusion of the key features from any viewpoint. The objects were generated by the Symbolics S-Geometry$^{TM}$ modeling package, and rendered with a visualization graphics tool (AVS, Stardent, Inc.). Each object consisted of seven connected equal-length segments, pointing in random directions and distributed equally around the origin (for further details, see Edelman and Bülthoff, 1990).

In the psychophysical experiments of Bülthoff and Edelman (1991), subjects were

shown a target wire from two standard views, located 75° apart along the equator of the viewing sphere. The target oscillated around each of the two standard orientations with an amplitude of ±15° about a fixed vertical axis, with views spaced at 3° increments. Test views were located either along the equator – on the minor arc bounded by the two standard views (INTER condition) or on the corresponding major arc (EXTRA condition) – or on the meridian passing through one of the standard views (ORTHO condition). Testing was conducted according to a two-alternative forced choice (2AFC) paradigm, in which subjects were asked to indicate whether the displayed image constituted a view of the target object shown during the preceding training session. Test images were either unfamiliar views of the training object, or random views of a distractor (one of a distinct set of objects generated by the same procedure).

To apply the above paradigm to the BCM network, the objects were rendered in a 63 × 63 array, at 8 bits/pixel, under simulated illumination that combined ambient lighting of relative strength 0.3 with a point source of strength 1.0 at infinity. The study described below involved six-way classification, which is more difficult than the 2AFC task used in the psychophysical experiments. The six wires used

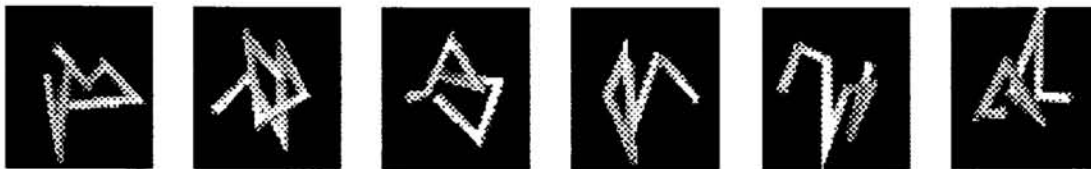

Figure 2: The six wires used in the computational experiments, as seen from a single view point.

in the experiments are depicted in Figure 2. Given the task of recognizing the six wires, the network extracted features that corresponded to small patches of the different images, namely areas that either remained relatively invariant under the rotation performed during training, or represented distinctive features of specific wires (Intrator and Gold, 1991). The classification results were in good agreement with the psychophysical data of Bülthoff and Edelman (1991): (1) the error rate was the lowest in the INTER condition, (2) recognition deteriorated to chance level with increased misorientation in the EXTRA and ORTHO conditions, and (3) horizontal training led to a better performance in the INTER condition than did vertical training.[1] The first two points were interpreted as resulting from the ability of the BCM network to extract rotation-invariant features. Indeed, features appearing in all the training views would be expected to correspond to the INTER condition. EXTRA and ORTHO views, on the other hand, are less familiar and therefore yield worse performance, and also may require features other than the rotation-invariant ones extracted by the model.

## 2    Examining the Features of Recognition

To understand the meaning of the features extracted by the BCM network under
the various conditions, and to establish a basis for further comparison between the
psychophysical experiments and computational models, we developed a method for
occluding key features from the images and examining the subsequent effects on the
various recognition tasks.

### 2.1    The Occlusion Experiment

In this experiment, some of the features previously extracted by the network could
be occluded during training and/or testing. Each input to a BCM neuron in our
model corresponds to a particular point in the 2D input image, while "features"
correspond to combinations of excitatory and inhibitory inputs. Assuming that in-
puts with strong positive weights constitute a significant proportion of the features,
we occluded (set to 0) input pixels whose previously computed synaptic weight ex-
ceeded a preset threshold. Figure 3 shows a synaptic weight matrix defining a set
of features, and the set of wires with the corresponding features occluded.

The main hypothesis we tested concerns the general utility of the extracted fea-
tures for recognition. If the features extracted by the BCM network do capture
rotation-invariant aspects of the object and can support recognition across a va-
riety of rotations, then occluding those features during training should lead to a
pronounced and general decline in recognition performance of the model. In par-
ticular, recognition should deteriorate most significantly in the INTER and EXTRA
cases, since they lie in the plane of rotation during training and therefore can be
expected to rely to a larger extent on rotation-invariant features. Little change
should be seen in the ORTHO condition, on the other hand, because recognition of
ORTHO views, situated outside the plane of rotation defined by the training phase,
does not benefit from rotation-invariant features.

### 2.2    Results and Discussion

When there was no occlusion, the pattern of the model's performance replicated
the results of the psychophysical experiments of (Bülthoff and Edelman, 1991).
Specifically, the best performance was achieved for INTER views, with progressive
deterioration under EXTRA and ORTHO conditions (Intrator and Gold, 1991; see
Figure 4). The results of simulations involving occlusion of key features during
training and no occlusion during testing are illustrated in Figure 5. Essentially
the same results were obtained when occlusion was done during either training or
testing.

Occlusion of the key features led to a number of interesting results. First, when
features in the training image were occluded, occluding the same features during
testing made little difference. This is not unexpected, since these features were not
used to build the internal representation of the objects. Second, there was a general
decline in performance within the plane of rotation used during training (especially
in the INTER condition) when the extracted features were occluded. This is a
strong indication that the features initially chosen by the network were in fact those
features which best described the object across a range of rotations. Third, there

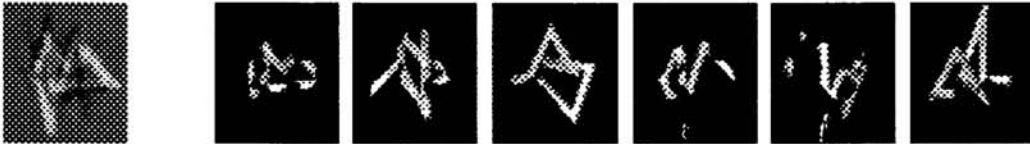

Figure 3: Wires occluded with a feature extracted by BCM network (left).

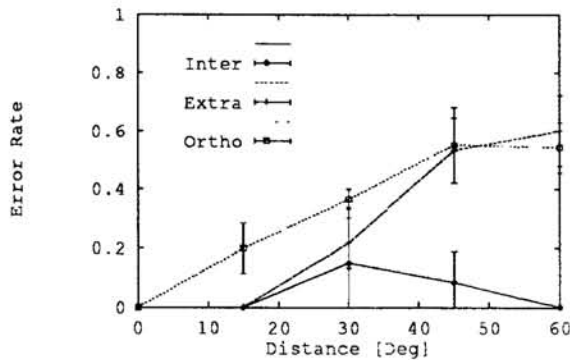

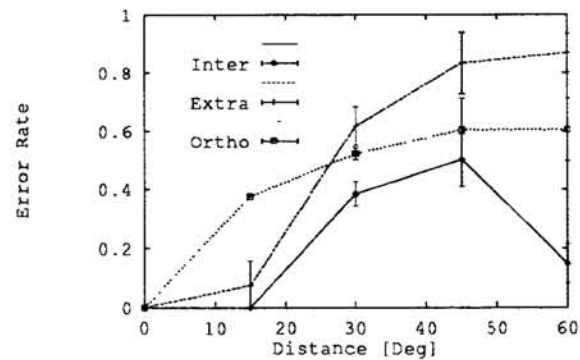

Figure 4: Misclassification performance, regular training.

Figure 5: Misclassification performance, training on occluded images.

was little degradation of performance in the ORTHO condition when features were occluded during training. This result lends further support to the notion that the extracted features emphasized rotation-invariant characteristics of the objects, as abstracted in the training phase. Finally, we mention that the occlusion of the same features in a new psychophysical experiment caused the same selective deterioration found in the simulations to appear in the human subjects' performance. Specifically, the subjects' error rate was elevated in the INTER condition more than in the other conditions, and this effect was significantly stronger for occlusion masks obtained from the extracted features than for other, randomized, masks (Sklar et al., 1991).

To summarize, this work was undertaken to elucidate the nature of the *features of recognition* of 3D objects. We were especially interested in the features extracted by an unsupervised BCM network, and in their relation to computational and psychophysical findings concerning object recognition. We compared recognition performance of our model following training that involved features extracted by the BCM network with performance in the absence of these features. We found that the model's performance was affected by the occlusion of key features in a manner consistent with their predicted computational role. This method of testing the relative importance of features has also been applied in psychophysical experiments. Preliminary results of those experiments show that feature-derived masks have a stronger effect on human performance compared to other masks that occlude the same proportion of the image, but are not obtained via the BCM model. Taken together, these results demonstrate the strength of the dimensionality reduction approach to feature extraction, and provide a foundation for examining the link

between computational and psychophysical studies of the features of recognition.

## Acknowledgements

Research was supported by the National Science Foundation, the Army Research Office, and the Office of Naval Research.

## Footnotes

[1] The horizontal-vertical asymmetry might be related to an asymmetric structure of the visual field in humans (Hughes, 1977). This asymmetry was modeled by increasing the resolution along the horizontal axis.

## References

Bienenstock, E. L., Cooper, L. N., and Munro, P. W. (1982). Theory for the development of neuron selectivity: orientation specificity and binocular interaction in visual cortex. *Journal Neuroscience*, 2:32–48.

Bülthoff, H. H. and Edelman, S. (1991). Psychophysical support for a 2D interpolation theory of object recognition. *Proceedings of the National Academy of Science*. to appear.

Duda, R. O. and Hart, P. E. (1973). *Pattern Classification and Scene Analysis.* John Wiley, New York.

Edelman, S. (1991). Features of recognition. CS-TR 10, Weizmann Institute of Science.

Edelman, S. and Bülthoff, H. H. (1990). Viewpoint-specific representations in three-dimensional object recognition. A.I. Memo No. 1239, Artificial Intelligence Laboratory, Massachusetts Institute of Technology.

Edelman, S., Bülthoff, H. H., and Sklar, E. (1991). Task and object learning in visual recognition. CBIP Memo No. 63, Center for Biological Information Processing, Massachusetts Institute of Technology.

Friedman, J. H. (1987). Exploratory projection pursuit. *Journal of the American Statistical Association*, 82:249–266.

Huber, P. J. (1985). Projection pursuit. (with discussion). *The Annals of Statistics*, 13:435–475.

Hughes, A. (1977). The topography of vision in mammals of contrasting live style: Comparative optics and retinal organisation. In Crescitelli, F., editor, *The Visual System in Vertebrates, Handbook of Sensory Physiology VII/5*, pages 613–756. Springer Verlag, Berlin.

Intrator, N. (1990). Feature extraction using an unsupervised neural network. In Touretzky, D. S., Ellman, J. L., Sejnowski, T. J., and Hinton, G. E., editors, *Proceedings of the 1990 Connectionist Models Summer School*, pages 310–318. Morgan Kaufmann, San Mateo, CA.

Intrator, N. (1992). Feature extraction using an unsupervised neural network. *Neural Computation*, 4:98–107.

Intrator, N. and Cooper, L. N. (1991). Objective function formulation of the BCM theory of visual cortical plasticity: Statistical connections, stability conditions. *Neural Networks*. To appear.

Intrator, N. and Gold, J. I. (1991). Three-dimensional object recognition of gray level images: The usefulness of distinguishing features. Submitted.

Intrator, N. and Tajchman, G. (1991). Supervised and unsupervised feature extraction from a cochlear model for speech recognition. In Juang, B. H., Kung, S. Y., and Kamm, C. A., editors, *Neural Networks for Signal Processing – Proceedings of the 1991 IEEE Workshop*, pages 460–469.

LaBerge, D. (1976). Perceptual learning and attention. In Estes, W. K., editor, *Handbook of learning and cognitive processes*, volume 4, pages 237–273. Lawrence Erlbaum, Hillsdale, New Jersey.

Poggio, T. and Edelman, S. (1990). A network that learns to recognize three-dimensional objects. *Nature*, 343:263–266.

Shepard, R. N. (1987). Toward a universal law of generalization for psychological science. *Science*, 237:1317–1323.

Sklar, E., Intrator, N., Gold, J. J., Edelman, S. Y., and Bülthoff, H. H. (1991). A hierarchical model for 3D object recognition based on 2D visual representation. In *Neurosci. Soc. Abs.*


